# Online Learning of Non-stationary Sequences

**Claire Monteleoni and Tommi Jaakkola**
MIT Computer Science and Artificial Intelligence Laboratory
200 Technology Square
Cambridge, MA 02139
{cmontel,tommi}@ai.mit.edu

## Abstract

We consider an online learning scenario in which the learner can make
predictions on the basis of a fixed set of experts. We derive upper and
lower relative loss bounds for a class of universal learning algorithms in-
volving a switching dynamics over the choice of the experts. On the basis
of the performance bounds we provide the optimal *a priori* discretiza-
tion for learning the parameter that governs the switching dynamics. We
demonstrate the new algorithm in the context of wireless networks.

## 1 Introduction

We focus on the online learning framework in which the learner has access to a set of ex-
perts but possesses no other *a priori* information relating to the observation sequence. In
such a scenario the learner may choose to quickly identify a single best expert to rely on
[12], or switch from one expert to another in response to perceived changes in the observa-
tion sequence [8], thus making assumptions about the switching dynamics. The ability to
shift emphasis from one "expert" to another, in response to changes in the observations, is
valuable in many applications, including energy management in wireless networks.

Many algorithms developed for universal prediction on the basis of a set of experts have
clear performance guarantees (e.g., [12, 6, 8, 14]). The performance bounds characterize
the regret relative to the best expert, or best sequence of experts, chosen in hindsight. Al-
gorithms with such relative loss guarantees have also been developed for adaptive game
playing [5], online portfolio management [7], paging [3] and the $k$-armed bandit problem
[1]. Other relative performance measures for universal prediction involve comparing across
systematic variations in the sequence [4].

Here we extend the class of algorithms considered in [8], by learning the switching-rate
parameter online, at the optimal resolution. Our goal of removing the switching-rate as a
parameter is similar to Vovk's in [14], though the approach and the comparison class for
the bounds differ. We provide upper and lower performance bounds, and demonstrate the
utility of these algorithms in the context of wireless networks.

## 2 Algorithms and performance guarantees

The learner has access to $n$ experts, $a_1, \ldots, a_n$, and each expert makes a prediction at each
time-step over a finite (known) time period $t = 1, \ldots, T$. We denote the $i^{th}$ expert at

time $t$ as $a_{i,t}$ to suppress any details about how the experts arrive at their predictions and what information is available to facilitate the predictions. These details may vary from one expert to another and may change over time. We denote the non-negative prediction loss of expert $i$ at time $t$ as $L(i, t)$, where the loss, a function of $t$, naturally depends on the observation $y_t \in \mathcal{Y}$ at time $t$. We consider here algorithms that provide a distribution $p_t(i)$, $i = 1, \ldots, n$, over the experts at each time point. The prediction loss of such an algorithm is denoted by $L(p_t, t)$.

For the purpose of deriving learning algorithms such as `Static-expert` and `Fixed-share` described in [8], we associate the loss of each expert with a predictive probability so that $-\log p(y_t | y_{t-1}, \ldots, y_1, i) = L(i, t)$. We define the loss of any probabilistic prediction to be the log-loss:

$$L(p_t, t) = -\log \sum_{i=1}^{n} p_t(i) \, p(y_t | i, y_1, \ldots, y_{t-1}) = -\log \sum_{i=1}^{n} p_t(i) e^{-L(i,t)} \qquad (1)$$

Many other definitions of the loss corresponding to $p_t(\cdot)$ can be bounded by a scaled log-loss [6, 8]. We omit such modifications here as they do not change the essential nature of the algorithms nor their analysis.

The algorithms combining expert predictions can be now derived as simple Bayesian estimation methods calculating the distribution $p_t(i) = P(i | y_1, \ldots, y_{t-1})$ over the experts on the basis of the observations seen so far. $p_1(i) = 1/n$ for any such method as any other initial bias could be detrimental in terms of relative performance guarantees. Updating $p_t(\cdot)$ involves assumptions about how the optimal choice of expert can change with time. For simplicity, we consider here only a Markov dynamics, defined by $p(i_t | i_{t-1}; \alpha)$, where $\alpha$ parameterizes the one-step transition probabilities. Allowing switches at rate $\alpha$, we define[1]

$$p(i_t | i_{t-1}; \alpha) = (1 - \alpha)\delta(i_t, i_{t-1}) + \frac{\alpha}{n - 1}[1 - \delta(i_t, i_{t-1})] \qquad (2)$$

which corresponds to the `Fixed-share` algorithm, and yields the `Static-expert` algorithm when $\alpha = 0$. The Bayesian algorithm updating $p_t(\cdot)$ is defined analogously to forward propagation in generalized HMMs (allowing observation dependence on past):

$$p_t(i; \alpha) = \frac{1}{Z_t} \sum_{j=1}^{n} p_{t-1}(j; \alpha) e^{-L(j, t-1)} p(i | j; \alpha) \qquad (3)$$

where $Z_t$ normalizes the distribution. While we have made various probabilistic assumptions (e.g., conditional independence of expert predictions) in deriving the algorithm, the algorithms can be used in a context where no such statistical assumptions about the observation sequence or the experts are warranted. The performance guarantees we provide below for these algorithms do not require these assumptions.

## 2.1 Relative loss bounds

The existing upper bound on the relative loss of the `Fixed-share` algorithm [8] is expressed in terms of the loss of the algorithm relative to the loss of the best $k$-partition of the observation sequence, where the best expert is assigned to each segment. We start by providing here a similar guarantee but characterizing the regret relative to the best `Fixed-share` algorithm, parameterized by $\alpha^*$, where $\alpha^*$ is chosen in hindsight after having seen the observation sequence. Our proof technique is different from [8] and gives rise to simple guarantees for a wider class of prediction methods, along with a lower bound on this regret.

**Lemma 1** *Let $L_T(\alpha) = \sum_{t=1}^{T} L(p_{t;\alpha}, t)$, $\alpha \in [0, 1]$, be the cumulative loss of the* `Fixed-share` *algorithm on an arbitrary sequence of observations. Then for any $\alpha, \alpha^*$:*

$$L_T(\alpha) - L_T(\alpha^*) = -\log \left[ E_{\hat{\alpha} \sim Q} \, e^{(T-1)[D(\hat{\alpha} \| \alpha^*) - D(\hat{\alpha} \| \alpha)]} \right] \tag{4}$$

**Proof**: The cumulative log-loss of the Bayesian algorithm can be expressed in terms of negative log-probability of all the observations:

$$L_T(\alpha) = -\log[\sum_{\vec{s}} \phi(\vec{s}) p(\vec{s}; \alpha)] \tag{5}$$

where $\vec{s} = \{i_1, \ldots, i_T\}$, $\phi(\vec{s}) = \prod_{t=1}^{T} e^{-L(i_t, t)}$, and $p(\vec{s}; \alpha) = p_1(i_1) \prod_{t=2}^{T} p(i_t | i_{t-1}; \alpha)$. Consequently, $L_T(\alpha) - L_T(\alpha^*)$

$$
\begin{aligned}
&= -\log \frac{\sum_{\vec{s}} \phi(\vec{s}) p(\vec{s}; \alpha)}{\sum_{\vec{r}} \phi(\vec{r}) p(\vec{r}; \alpha^*)} = -\log \left[ \sum_{\vec{s}} \left( \frac{\phi(\vec{s}) p(\vec{s}; \alpha^*)}{\sum_{\vec{r}} \phi(\vec{r}) p(\vec{r}; \alpha^*)} \right) \frac{p(\vec{s}; \alpha)}{p(\vec{s}; \alpha^*)} \right] \\
&= -\log \left[ \sum_{\vec{s}} Q(\vec{s}; \alpha^*) \frac{p(\vec{s}; \alpha)}{p(\vec{s}; \alpha^*)} \right] = -\log \left[ \sum_{\vec{s}} Q(\vec{s}; \alpha^*) e^{\log \frac{p(\vec{s}; \alpha)}{p(\vec{s}; \alpha^*)}} \right] \\
&= -\log \left[ \sum_{\vec{s}} Q(\vec{s}; \alpha^*) e^{(T-1)\left( \hat{\alpha}(\vec{s}) \log \frac{\alpha}{\alpha^*} + (1-\hat{\alpha}(\vec{s})) \log \frac{1-\alpha}{1-\alpha^*} \right)} \right]
\end{aligned}
$$

where $Q(\vec{s}; \alpha^*)$ is the posterior probability over the choices of experts along the sequence, induced by the hindsight-optimal switching-rate $\alpha^*$, and $\hat{\alpha}(\vec{s})$ is the empirical fraction of non-self-transitions in the selection sequence $\vec{s}$. This can be rewritten as the expected value of $\hat{\alpha}$ under distribution $Q$. $\square$

We obtain upper and lower bounds on regret by optimizing $Q$ in $\mathcal{Q}$, the set of all distributions over $\hat{\alpha} \in [0, 1]$, of the expression for regret.

### 2.1.1 Upper bound

The upper bound follows from solving: $\max_{Q \in \mathcal{Q}} \left\{ -\log \left[ E_{\hat{\alpha} \sim Q} \, e^{(T-1)[D(\hat{\alpha} \| \alpha^*) - D(\hat{\alpha} \| \alpha)]} \right] \right\}$ subject to the constraint that $\alpha^*$ has to be the hindsight-optimal switching-rate, i.e. that:
(C1)   $\frac{d}{d\alpha}(L_T(\alpha) - L_T(\alpha^*))_{|\alpha=\alpha^*} = 0$

**Theorem 1** *Let $L_T(\alpha^*) = \min_\alpha L_T(\alpha)$ be the loss of the best* `Fixed-share` *algorithm chosen in hindsight. Then for any $\alpha \in [0, 1]$, $L_T(\alpha) - L_T(\alpha^*) \le (T-1) D(\alpha^* \| \alpha)$, where $D(\alpha^* \| \alpha)$ is the relative entropy between Bernoulli distributions defined by $\alpha^*$ and $\alpha$.*

The bound vanishes when $\alpha = \alpha^*$ and does not depend directly on the number of experts. The dependence on $n$ may appear indirectly through $\alpha^*$, however. While the regret appears proportional to $T$, this dependence vanishes for any reasonable learning algorithm that is guaranteed to find $\alpha$ such that $D(\alpha^* \| \alpha) \le \mathcal{O}(1/T)$, as we will show in Section 3.

Theorem 1 follows, as a special case, from an analogous result for algorithms based on arbitrary first-order Markov transition dynamics. In the general case, the regret bound is: $(T-1) \max_i D(P(j|i, \alpha^*) \| P(j|i, \alpha))$, where $\alpha, \alpha^*$ are now transition matrices, and $D(\cdot \| \cdot)$ is the relative entropy between discrete distributions. For brevity, we provide only the proof of the scalar case of Theorem 1.

**Proof**: Constraint (C1) can be expressed simply as $\frac{d}{d\alpha} L_T(\alpha)_{|\alpha=\alpha^*} = 0$, which is equivalent to $E_{\hat{\alpha} \sim Q}\{\hat{\alpha}\} = \alpha^*$. Taking the expectation outside the logarithm, in Equation 4, results in the upper bound. $\square$

### 2.1.2 Lower bound

The relative losses obviously satisfy $L_T(\alpha) - L_T(\alpha^*) \geq 0$ providing a trivial lower bound. Any non-trivial lower bound on the regret cannot be expressed only in terms of $\alpha$ and $\alpha^*$, but needs to incorporate some additional information about the losses along the observation sequence. We express the lower bound on the regret as a function of the relative quality $\beta^*$ of the minimum $\alpha^*$:

$$\beta^* = \frac{\alpha^*(1 - \alpha^*)}{T - 1} \frac{d^2}{d\alpha^2} L_T(\alpha)_{|\alpha = \alpha^*} \tag{6}$$

where the normalization guarantees that $\beta^* \leq 1$. $\beta^* \geq 0$ for any $\alpha^*$ that minimizes $L_T(\alpha)$.

The lower bound is found by solving: $\min_{Q \in \mathcal{Q}} \left\{ -\log \left[ E_{\hat{\alpha} \sim Q} \, e^{(T-1)[D(\hat{\alpha}\|\alpha^*) - D(\hat{\alpha}\|\alpha)]} \right] \right\}$ subject to both constraint (C1) and (C2) $\quad \frac{d^2}{d\alpha^2}(L_T(\alpha) - L_T(\alpha^*))_{|\alpha = \alpha^*} = \frac{\beta^*(T-1)}{\alpha^*(1-\alpha^*)}$

**Theorem 2** *Let $\beta^*$ and $\alpha^*$ be defined as above based on an arbitrary observation sequence, and $q_1 = [1 + \frac{T-1}{1-\beta^*} \frac{1-\alpha^*}{\alpha^*}]^{-1}$ and $q_0 = [1 + \frac{T-1}{1-\beta^*} \frac{\alpha^*}{1-\alpha^*}]^{-1}$. Then*

$$L_T(\alpha) - L_T(\alpha^*) \geq -\log \left[ E_{\hat{\alpha} \sim Q} \, e^{(T-1)[D(\hat{\alpha}\|\alpha^*) - D(\hat{\alpha}\|\alpha)]} \right] \tag{7}$$

*where $Q(1) = q_1$ and $Q((\alpha^* - q_1)/(1 - q_1)) = 1 - q_1$ whenever $\alpha \geq \alpha^*$; $Q(0) = q_0$ and $Q(\alpha^*/(1 - q_0)) = 1 - q_0$ otherwise.*

Proof omitted due to space constraints. The upper and lower bounds agree for all $\alpha, \alpha^* \in (0, 1)$ when $\beta^* \to 1$. Thus there may exist observation sequences on which `Fixed-share`, using $\alpha \neq \alpha^*$, must incur regret linear in $T$.

## 2.2 Algorithm `Learn-`$\alpha$

We now give an algorithm to learn the switching-rate simultaneously to updating the probability weighting over the experts. Since the cumulative loss $L_t(\alpha)$ of each `Fixed-share` algorithm running with switching parameter $\alpha$ can be interpreted as a negative log-probability, the posterior distribution over the switching-rate becomes

$$p_t(\alpha) = P(\alpha|y_{t-1}, \ldots, y_1) \propto e^{-L_{t-1}(\alpha)} \tag{8}$$

assuming a uniform prior over $\alpha \in [0, 1]$. As a predictive distribution $p_t(\alpha)$ does not include the observation at the same time point. We can view this algorithm as finding the single best "$\alpha$-expert," where the collection of $\alpha$-experts is given by `Fixed-share` algorithms running with different switching-rates, $\alpha$.

We will consider a finite resolution version of this algorithm, allowing only $m$ possible choices for the switching-rate, $\{\alpha_1, \ldots, \alpha_m\}$. For a sufficiently large $m$ and appropriately chosen values $\{\alpha_j\}$, we expect to be able to always find $\alpha_j \approx \alpha^*$ and suffer only a minimal additional loss due to not being able to represent the hindsight-optimal value exactly.

Let $p_{t,j}(i)$ be the distribution over experts defined by the $j^{th}$ `Fixed-share` algorithm corresponding to $\alpha_j$, and let $p_t^{top}(j)$ be the top-level algorithm producing a weighting over such `Fixed-share` experts. The top-level algorithm is given by

$$p_t^{top}(j) = \frac{1}{Z_t} p_{t-1}^{top}(j) e^{-L(p_{t-1,j}, t-1)} \tag{9}$$

where $p_1^{top}(j) = 1/m$, and the loss per time-step becomes

$$L^{top}(p_t^{top}, t) = -\log \sum_{j=1}^{m} p_t^{top}(j) e^{-L(p_{t,j}, t)} = -\log \sum_{j=1}^{m} \sum_{i=1}^{n} p_t^{top}(j) p_{t,j}(i) e^{-L(i,t)} \tag{10}$$

as is appropriate for a hierarchical Bayesian method.

# 3 Relative loss and optimal discretization

We derive here the optimal choice of the discrete set $\{\alpha_1, \ldots, \alpha_m\}$ on the basis of the upper bound on relative loss. We begin by extending Theorem 1 to provide an analogous guarantee for the `Learn-`$\alpha$ algorithm.

**Corollary to Theorem 1** *Let $L_T^{top}$ be the cumulative loss of the hierarchical* `Learn-`$\alpha$ *algorithm using* $\{\alpha_1, \ldots, \alpha_m\}$. *Then*

$$L_T^{top} - L_T(\alpha^*) \leq \log(m) + (T-1) \min_{j=1,\ldots,m} D(\alpha^* \| \alpha_j) \tag{11}$$

The hierarchical algorithm involves two competing goals that manifest themselves in the regret: 1) the ability to identify the best `Fixed-share` expert, which degrades for larger $m$, and 2) the ability to find $\alpha_j$ whose loss is close to the optimal $\alpha$ for that sequence, which improves for larger $m$. The additional regret arising from having to consider a number of non-optimal values of the parameter, in the search, comes from the relative loss bound for the `Static-Expert` algorithm, i.e. the relative loss associated with tracking the best single expert [8, 12]. This regret is simply $\log(m)$ in our context. More precisely, the corollary follows directly from successive application of that single expert relative loss bound, and then our `Fixed-share` relative loss bound (Theorem 1):

$$L_T^{top} - L_T(\alpha^*) \leq \log(m) + \min_{j=1,\ldots,m} L_T(\alpha_j) \tag{12}$$

$$\leq \log(m) + (T-1) \min_{j=1,\ldots,m} D(\alpha^* \| \alpha_j) \tag{13}$$

## 3.1 Optimal discretization

We start by finding the smallest discrete set of switching-rate parameters so that any additional regret due to discretization does not exceed $(T-1)\delta$, for some threshold $\delta$. In other words, we find $m = m(\delta)$ values $\alpha_1, \ldots, \alpha_{m(\delta)}$ such that

$$\max_{\alpha^* \in [0,1]} \min_{j=1,\ldots,m(\delta)} D(\alpha^* \| \alpha_j) = \delta \tag{14}$$

The resulting discretization, a function of $\delta$, can be found algorithmically as follows. First, we set $\alpha_1$ so that $\max_{\alpha^* \in [0,\alpha_1]} D(\alpha^* \| \alpha_1) = D(0 \| \alpha_1) = \delta$ implying that $\alpha_1 = 1 - e^{-\delta}$. Each subsequent $\alpha_j$ is found conditionally on $\alpha_{j-1}$ so that

$$\max_{\alpha^* \in [\alpha_{j-1}, \alpha_j]} \min\{D(\alpha^* \| \alpha_{j-1}), D(\alpha^* \| \alpha_j)\} = \delta \tag{15}$$

The maximizing $\alpha^*$ can be solved explicitly by equating the two relative entropies giving

$$\alpha^* = \log\left(\frac{1 - \alpha_{j-1}}{1 - \alpha_j}\right) \left( \log\left(\frac{\alpha_j}{\alpha_{j-1}} \frac{1 - \alpha_{j-1}}{1 - \alpha_j}\right) \right)^{-1} \tag{16}$$

which lies within $[\alpha_{j-1}, \alpha_j]$ and is an increasing function of the new point $\alpha_j$. Substituting this $\alpha^*$ back into one of the relative entropies we can set $\alpha_j$ so that $D(\alpha^* \| \alpha_{j-1}) = \delta$. The relative entropy is an increasing function of $\alpha_j$ (through $\alpha^*$) and the solution is obtained easily via, e.g., bisection search. The iterative procedure of generating new values $\alpha_j$ can be stopped after the new point exceeds $1/2$; the remaining levels can be filled-in by symmetry so long as we also include $1/2$. The resulting discretization is not uniform but denser towards the edges; the spacing around the edges is $\mathcal{O}(\delta)$, and $\mathcal{O}(\sqrt{\delta})$ around $1/2$.

For small values of $\delta$, the logarithm of the number of resulting discretization levels, or $\log m(\delta)$, closely approximates $-1/2 \log \delta$. We can then optimize the regret bound (11): $-1/2 \log \delta + (T-1)\delta$, yielding $\delta^* = 1/(2T)$, and $m(\delta^*) = \sqrt{2T}$. Thus we will need $\mathcal{O}(\sqrt{T})$ settings of $\alpha$, as in the case of choosing the levels uniformly with spacing $\sqrt{\delta}$. The uniform discretization would not, however, possess the same regret guarantee, resulting in a higher than necessary loss due to discretization.

### 3.1.1 Optimized regret bound for Learn-$\alpha$

The optimized regret bound for Learn-$\alpha(\delta^*)$ is thus (approximately) $\frac{1}{2}\log T + c$, which is comparable to analysis of universal coding for word-length $T$ [11]. The optimal discretization for learning the parameter is not affected by $n$, the number of original experts. Unlike regret bounds for Fixed-share, the value of the bound does not depend on the observation sequence. And notably, in comparison to the lower bound on Fixed-share's performance, Learn-$\alpha$'s regret is at most logarithmic in $T$.

## 4 Application to wireless networks

We applied the Learn-$\alpha$ algorithm to an open problem in computer networks: managing the tradeoff between energy consumption and performance in wireless nodes of the IEEE 802.11 standard [9]. Since a node cannot receive packets while asleep, yet maintaining the awake state drains energy, the existing standard uses a fixed polling time at which a node should wake from the sleep state to poll its neighbors for buffered packets. Polling at fixed intervals however, does not respond optimally to current network activity. This problem is clearly an appropriate application for an online learning algorithm, such as Fixed-share due to [8]. Since we are concerned with wireless, mobile nodes, there is no principled way to set the switching-rate parameter *a priori*, as network activity varies not only over time, but across location, and the location of the mobile node is allowed to change. We can therefore expect an additional benefit from learning the switching-rate.

Previous work includes Krashinsky and Balakrishnan's [10] Bounded Slowdown algorithm which uses an adaptive control loop to change polling time based on network conditions. This algorithm uses parameterized exploration intervals, and the tradeoff is not managed optimally. Steinbach applied reinforcement learning [13] to this problem, yet required an unrealistic assumption: that network activity possesses the Markov property.

We instantiate the experts as deterministic algorithms assuming constant polling times. Thus we use $n$ experts, each corresponding to a different but fixed polling time in milliseconds ($ms$): $T_i : i \in \{1 \dots n\}$ The experts form a discretization over the range of possible polling times. We then apply the Learn-$\alpha$ algorithm exactly as in our previous exposition, using the discretization defined by $\delta^*$, and thus running $m(\delta^*)$ sub-algorithms, each running Fixed-share with a different $\alpha_j$. In this application, the learning algorithm can only receive observations, and perform learning updates, when it is awake. So our subscript $t$ here signifies only wake times, not every time epoch at which bytes might arrive.

We define the loss function, $L$, to reflect the tradeoff inherent in the conflicting goals of minimizing both the energy usage of the node, and the network latency it introduces by sleeping. We propose a loss function that is one of many functions proportional to this tradeoff. We define loss per expert $i$ as:

$$Loss(i,t) = \gamma \frac{I_t T_i^2}{2T_t} + \frac{1}{T_i} \qquad (17)$$

where $I_t$ is the observation the node receives, of how many bytes arrived upon awakening at time $t$, and $T_t$ is the length of time that the node just slept. The first term models the average latency introduced into the network by buffering those bytes, and scales $I_t$ to the number of bytes that would have arrived had the node slept for time $T_i$ instead of $T_t$, under the assumption that the bytes arrived at a uniform rate. The second term models the energy consumption of the node, based on the design that the node wakes only after an interval $T_t$ to poll for buffered bytes, and the fact that it consumes less energy when asleep than awake. The objective function is a sum of convex functions and thus admits a unique minimum. $\gamma > 0$ allows for scaling between the units of information and time, and the ability to encode a preference for the ratio between energy and latency that the user favors.

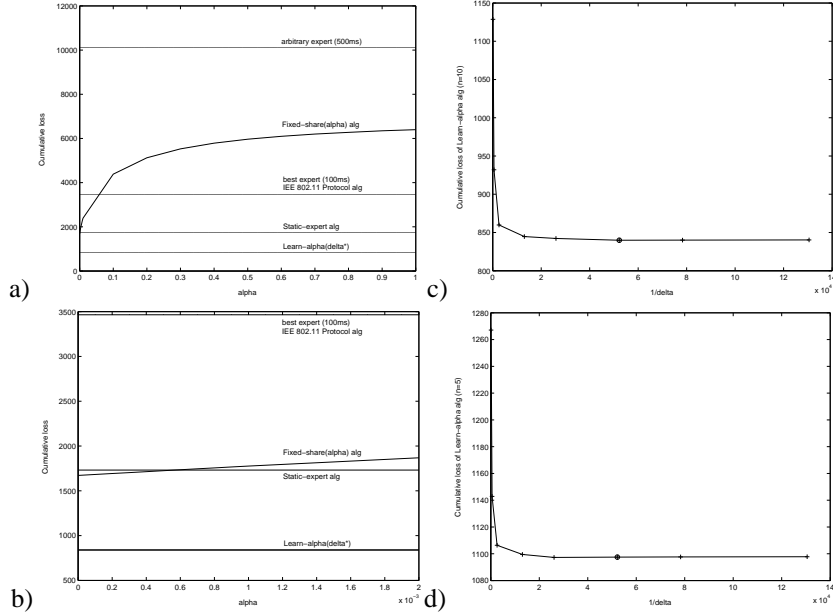

Figure 1: a) Cumulative loss of `Fixed-share(α)` as a function of $\alpha$, compared to the cumulative loss on the same trace of the 802.11 protocol, `Static-expert`, and `Learn-α(δ*)`. Figure b) zooms in on the first 0.002 of the $\alpha$ range. c) Cumulative loss of `Learn-α(δ)`, as a function of $1/\delta$, when $n = 10$, and b) $n = 5$. Circles at $1/\delta^* = 2\mathcal{T}$.

### 4.0.2 Experiments

We used traces of real network activity from [2], a UC Berkeley home dial-up server that monitored users accessing HTTP files from home. Multiple overlapping connections, passing through a collection node over several days, were recorded by start and end times, and number of bytes transferred. Per connection we smoothed the total number of bytes uniformly over $10ms$ intervals spanning its duration. We set $\gamma = 1.0 \times 10^{-7}$, calibrated to attain polling times within the range of the existing protocol.

Figure 1a) and b) compare cumulative loss of the various algorithms on a 4 hour trace, with observation epochs every $10ms$. This corresponds to approximately 26,100 training iterations for the learning algorithms. In the typical online learning scenario, $\mathcal{T}$, the number of learning iterations, i.e. the time horizen parameter to the loss bounds, is just the number of observation epochs. In this application, the number of training epochs need not match the number of observation epochs, since the application involves sleeping during many observation epochs, and learning is only done upon awakening. Since in these experiments the performance of the three learning algorithms are compared by each algorithm using $n$ experts spanning the range of $1000ms$ at regularly spaced intervals of $100ms$, to obtain a prior estimate of $\mathcal{T}$, we assume a mean sleep interval of $550ms$, the mean of the experts.

The `Static-expert` algorithm achieved lower cumulative loss than the best expert, since it can attain the optimal smoothed value over the desired range of polling times, whereas the expert values just form a discretization. On this trace, the optimal $\alpha$ for `Fixed-share` turns out to be extremely low. So for most settings of $\alpha$, one would be better off using a `Static-expert` model, yet as the second graph shows, there is a value of $\alpha$ below which it is beneficial to use `Fixed-share`. This lends validity to our fundamental goal of being able to quantify the level of non-stationarity of a process, in order

to better model it. Moreover there is a clear advantage to using Learn-$\alpha$, since without prior knowledge of the stochastic process to be observed, there is no optimal way to set $\alpha$.

Figure 1c) and d) show the cumulative loss of Learn-$\alpha$ as a function of $1/\delta$. We see that choosing $\delta = \frac{1}{27}$, matches the point in the curve beyond which one cannot significantly reduce cumulative loss by decreasing $\delta$. As expected, the performance of the algorithm levels off after the optimal $\delta$ that we can compute *a priori*. Our results also verify that the optimal $\delta$ is not significantly affected by the number of experts $n$.

## 5    Conclusion

We proved upper and lower bounds on the regret for a class of online learning algorithms, applicable to any sequence of observations. The bounds extend to richer models of non-stationary sequences, allowing the switching dynamics to be governed by an arbitrary transition matrix. We derived the regret-optimal discretization (including the overall resolution) for learning the switching-rate parameter in a simple switching dynamics, yielding an algorithm with stronger guarantees than previous algorithms. We exemplified the approach in the context of energy management in wireless networks. In future work, we hope to extend the online estimation of $\alpha$ and the optimal discretization to learning a full transition matrix.

## Footnotes

[1]where $\delta(\cdot, \cdot)$ is the Kronecker delta.

## References

[1] P. Auer, N. Cesa-Bianchi, Y. Freund, and R. E. Schapire. Gambling in a rigged casino: the adversarial multi-armed bandit problem. In *Proc. of the 36th Annual Symposium on Foundations of Computer Science*, pages 322–331, 1995.

[2] Berkeley. UC Berkeley home IP web traces. In *http://ita.ee.lbl.gov/html/contrib/UCB.home-IP-HTTP.html*, 1996.

[3] A. Blum, C. Burch, and A. Kalai. Finely-competitive paging. In *IEEE 40th Annual Symposium on Foundations of Computer Science*, page 450, New York, New York, October 1999.

[4] D. P. Foster and R. Vohra. Regret in the on-line decision problem. *Games and Economic Behavior*, 29:7–35, 1999.

[5] Y. Freund and R. Schapire. Adaptive game playing using multiplicative weights. *Games and Economic Behavior*, 29:79–103, 1999.

[6] D. Haussler, J. Kivinen, and M. K. Warmuth. Sequential prediction of individual sequences under general loss functions. *IEEE Trans. on Information Theory*, 44(5):1906–1925, 1998.

[7] D. P. Helmbold, R. E. Schapire, Y. Singer, and M. K. Warmuth. On-line portfolio selection using multiplicative updates. In *International Conference on Machine Learning*, pages 243–251, 1996.

[8] M. Herbster and M. K. Warmuth. Tracking the best expert. *Machine Learning*, 32:151–178, 1998.

[9] IEEE. Computer society LAN MAN standards committee. In *IEEE Std 802.11: Wireless LAN Medium Access Control and Physical Layer Specifications*, August 1999.

[10] R. Krashinsky and H. Balakrishnan. Minimizing energy for wireless web access with bounded slowdown. In *MobiCom 2002*, Atlanta, GA, September 2002.

[11] R. Krichevsky and V. Trofimov. The performance of universal encoding. *IEEE Trans. on Information Theory*, 27(2):199–207, 1981.

[12] N. Littlestone and M. K. Warmuth. The weighted majority algorithm. In *IEEE Symposium on Foundations of Computer Science*, pages 256–261, 1989.

[13] C. Steinbach. A reinforcement-learning approach to power management. In *AI Technical Report, M.Eng Thesis*, Artificial Intelligence Laboratory, MIT, May 2002.

[14] V. Vovk. Derandomizing stochastic prediction strategies. *Machine Learning*, 35:247–282, 1999.
